# Bayesian Regularization and Nonnegative Deconvolution for Time Delay Estimation

**Yuanqing Lin, Daniel D. Lee**
GRASP Laboratory, Department of Electrical and System Engineering
University of Pennsylvania, Philadelphia, PA 19104
*linyuanq, ddlee@seas.upenn.edu*

## Abstract

Bayesian Regularization and Nonnegative Deconvolution (BRAND) is proposed for estimating time delays of acoustic signals in reverberant environments. Sparsity of the nonnegative filter coefficients is enforced using an $L_1$-norm regularization. A probabilistic generative model is used to simultaneously estimate the regularization parameters and filter coefficients from the signal data. Iterative update rules are derived under a Bayesian framework using the Expectation-Maximization procedure. The resulting time delay estimation algorithm is demonstrated on noisy acoustic data.

## 1 Introduction

Estimating the time difference of arrival is crucial for binaural acoustic sound source localization[1]. A typical scenario is depicted in Fig. 1 where the azimuthal angle $\phi$ to the sound source is determined by the difference in direct propagation times of the sound to the two microphones. The standard signal processing algorithm for determining the time delay between two signals $s(t)$ and $x(t)$ relies upon computing the cross-correlation function[2]: $C(\Delta t) = \int dt\, x(t)s(t - \Delta t)$ and determining the time delay $\Delta t$ that maximizes the cross-correlation. In the presence of uncorrelated white noise, this procedure is equivalent to the optimal matched filter for detection of the time delayed signal.

However, a typical room environment is reverberant and the measured signal is contaminated with echoes from multiple paths as shown in Fig. 1. In this case, the cross-correlation and related algorithms may not be optimal for estimating the time delays. An alternative approach would be to estimate the multiple time delays as a linear deconvolution problem:

$$\min_{\boldsymbol{\alpha}} \|x(t) - \sum_i \alpha_i s(t - \Delta t_i)\|^2 \tag{1}$$

Unfortunately, this deconvolution can be ill-conditioned resulting in very noisy solutions for the coefficients $\boldsymbol{\alpha}$. Recently, we proposed incorporating nonnegativity constraints $\boldsymbol{\alpha} \geq 0$ in the deconvolution to overcome the ill-conditioned linear solutions [3]. The use of these constraints is justified by acoustic models that describe the theoretical room impulse response with nonnegative filter coeffients [4]. The resulting optimization problem can be written as the nonnegative quadratic programming problem:

$$\min_{\boldsymbol{\alpha} \geq 0} \|\mathbf{x} - \mathbf{S}\boldsymbol{\alpha}\|^2 \tag{2}$$

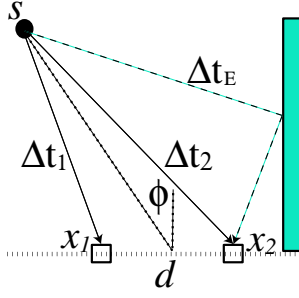

Figure 1: The typical scenario of reverberant signal. $x_2(t)$ comes from the direct path ($\Delta t_2$) and echo paths($\Delta t_E$).

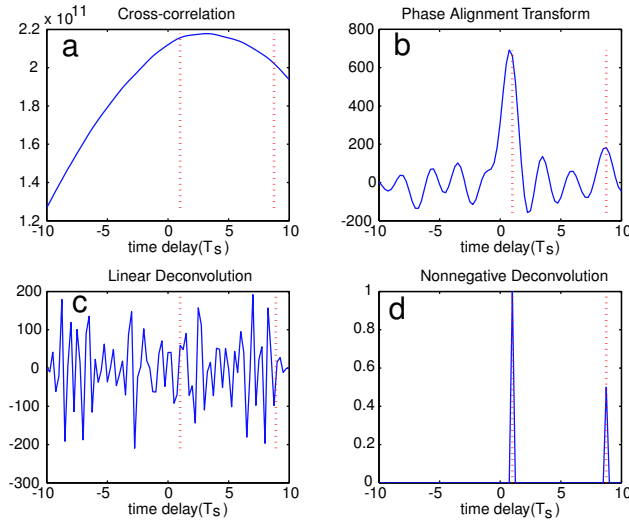

Figure 2: Time delay estimation of a speech signal with a) cross-correlation, b) phase alignment transform, c) linear deconvolution, d) nonnegative deconvolution. The observed signal $x(t) = s(t - T_s) + 0.5s(t - 8.75T_s)$ contains an additional time-delayed echo. $T_s$ is the sampling interval.

where $\mathbf{x} = \{x(t_1)\ x(t_2)\ \ldots\ x(t_N)\}^T$ is a $N \times 1$ data vector, $\mathbf{S} = \{s(t - \Delta t_1)\ s(t - \Delta t_2)\ \ldots\ s(t - \Delta t_M)\}$ is an $N \times M$ matrix, and $\boldsymbol{\alpha}$ is a $M \times 1$ vector of nonnegative coefficients.

Figure 2 compares the performance of cross-correlation, phase alignment transform(a generalized cross-correlation algorithm), linear deconvolution, and nonnegative deconvolution for estimating the time delays in a clean speech signal containing an echo. From the structure of the estimated coefficients, it is clear that nonnegative deconvolution can successfully discover the structure of the time delays present in the signal. However, in the presence of large background noise, it may be necessary to regularize the nonnegative quadratic optimization to prevent overfitting. In this case, we propose using an $L_1$-norm regularization to favor sparse solutions [5]:

$$\min_{\boldsymbol{\alpha} \geq 0} \|\mathbf{x} - \mathbf{S}\boldsymbol{\alpha}\|^2 + \hat{\lambda} \sum_i \alpha_i \tag{3}$$

In this formula, the parameter $\hat{\lambda}\ (\hat{\lambda} \geq 0)$ describes the trade-off between fitting the observed

data and enforcing sparse solutions. The proper choice of this parameter may be crucial in obtaining the optimal time delay estimates. In the rest of this manuscript, we introduce a proper generative model for these regularization parameters and filter coefficients within a probabilistic Bayesian framework. We show how these parameters can be efficiently determined using appropriate iterative estimates. We conclude by demonstrating and discussing the performance of our algorithm on noisy acoustic signals in reverberant environments.

## 2   Bayesian regularization

Instead of arbitrarily setting values for the regularization parameters, we show how a Bayesian framework can be used to automatically estimate the correct values from the data. Bayesian regularization has previously been successfully applied to neural network learning [6], model selection, and relevance vector machine (RVM) [7]. In these works, the fitting coefficients are assumed to have Gaussian priors, which lead to an $L_2$-norm regularization. In our model, we use $L_1$-norm sparsity regularization, and Bayesian framework will be used to optimally determine the appropriate regularization parameters.

Our probabilistic model assumes the observed data signal is generated by convolving the source signal with a nonnegative filter describing the room impulse response. This signal is then contaminated by additive Gaussian white noise with zero-mean and covariance $\sigma^2$:

$$P(\mathbf{x}|\mathbf{S}, \boldsymbol{\alpha}, \sigma^2) = \frac{1}{(2\pi\sigma^2)^{N/2}} \exp\left(-\frac{1}{2\sigma^2}\|\mathbf{x} - \mathbf{S}\boldsymbol{\alpha}\|^2\right). \tag{4}$$

To enforce sparseness in the filter coefficients $\boldsymbol{\alpha}$, an exponential prior distribution is used. This prior only has support in the nonnegative orthant and the sharpness of the distribution is given by the regularization parameter $\lambda$:

$$P(\boldsymbol{\alpha}|\lambda) = \lambda^M \exp\{-\lambda \sum_{i=1}^{M} \alpha_i\}, \quad \boldsymbol{\alpha} \geq 0. \tag{5}$$

In order to infer the optimal settings of the regularization parameters $\sigma^2$ and $\lambda$, Bayes rule is used to maximize the posterior distribution:

$$P(\lambda, \sigma^2|\mathbf{x}, \mathbf{S}) = \frac{P(\mathbf{x}|\lambda, \sigma^2, \mathbf{S})P(\lambda, \sigma^2)}{P(\mathbf{x}|\mathbf{S})}. \tag{6}$$

Assuming that $P(\lambda, \sigma^2)$ is relatively flat [8], estimating $\sigma^2$ and $\lambda$ is then equivalent to maximizing the likelihood:

$$P(\mathbf{x}|\lambda, \sigma^2, \mathbf{S}) = \frac{\lambda^M}{(2\pi\sigma^2)^{N/2}} \int_{\boldsymbol{\alpha} \geq 0} d\boldsymbol{\alpha} \, \exp[-F(\boldsymbol{\alpha})] \tag{7}$$

where

$$F(\boldsymbol{\alpha}) = \frac{1}{2\sigma^2}(\mathbf{x} - \mathbf{S}\boldsymbol{\alpha})^T(\mathbf{x} - \mathbf{S}\boldsymbol{\alpha}) + \lambda e^T \boldsymbol{\alpha} \tag{8}$$

and $\mathbf{e} = [1 \ 1 \ \ldots 1]^T$.

Unfortunately, the integral in Eq. 7 cannot be directly maximized. Previous approaches to Bayesian regularization have used iterative updates heuristically derived from self-consistent fixed point equations. In our model, the following iterative update rules for $\lambda$ and $\sigma^2$ can be derived using Expectation-Maximization:

$$\frac{1}{\lambda} \quad \longleftarrow \quad \frac{1}{M} \int_{\boldsymbol{\alpha} \geq 0} d\boldsymbol{\alpha} \, \mathbf{e}^T \boldsymbol{\alpha} Q(\boldsymbol{\alpha}) \tag{9}$$

$$\sigma^2 \quad \longleftarrow \quad \frac{1}{N} \int_{\boldsymbol{\alpha} \geq 0} d\boldsymbol{\alpha} \, (\mathbf{x} - \mathbf{S}\boldsymbol{\alpha})^T(\mathbf{x} - \mathbf{S}\boldsymbol{\alpha}) Q(\boldsymbol{\alpha}) \tag{10}$$

where the expectations are taken over the distribution

$$Q(\boldsymbol{\alpha}) = \frac{\exp[-F(\boldsymbol{\alpha})]}{\mathcal{Z}_{\boldsymbol{\alpha}}} , \tag{11}$$

with normalization $\mathcal{Z}_{\boldsymbol{\alpha}} = \int_{\boldsymbol{\alpha} \geq 0} d\boldsymbol{\alpha} \exp[-F(\boldsymbol{\alpha})]$. These updates have guaranteed convergence properties and can be intuitively understood as iteratively reestimating $\lambda$ and $\sigma^2$ based upon appropriate expectations over the current estimate for $Q(\boldsymbol{\alpha})$.

## 2.1 Estimation of $\boldsymbol{\alpha}^{ML}$

The integrals in Eqs. 9–10 are dominated by $\boldsymbol{\alpha} \approx \boldsymbol{\alpha}^{ML}$ where the most likely $\boldsymbol{\alpha}^{ML}$ is given by:

$$\boldsymbol{\alpha}^{ML} = \arg\min_{\boldsymbol{\alpha} \geq 0} \frac{1}{2\sigma^2}(\mathbf{x} - \mathbf{S}\boldsymbol{\alpha})^T(\mathbf{x} - \mathbf{S}\boldsymbol{\alpha}) + \boldsymbol{\lambda}^T\boldsymbol{\alpha}. \tag{12}$$

This optimization is equivalent to the nonnegative quadratic programming problem in Eq. 3 with $\hat{\lambda} = \lambda\sigma^2$. To efficiently compute $\boldsymbol{\alpha}^{ML}$, we have recently developed two distinct methods for optimizing Eq. 12.

The first method is based upon a multiplicative update rule for nonnegative quadratic programming [9]. We first write the problem in the following form:

$$\min_{\boldsymbol{\alpha} \geq 0} \frac{1}{2}\boldsymbol{\alpha}^T\mathbf{A}\boldsymbol{\alpha} + \mathbf{b}^T\boldsymbol{\alpha}, \tag{13}$$

where $\mathbf{A} = \frac{1}{\sigma^2}\mathbf{S}^T\mathbf{S}$, and $\mathbf{b} = -\frac{1}{\sigma^2}\mathbf{S}^T\mathbf{x}$.

First, we decompose the matrix $A = A^+ - A^-$ into its positive and negative components such that:

$$A_{ij}^+ = \begin{cases} A_{ij} & if \quad A_{ij} > 0 \\ 0 & if \quad A_{ij} \leq 0 \end{cases} \qquad A_{ij}^- = \begin{cases} 0 & if \quad A_{ij} \geq 0 \\ -A_{ij} & if \quad A_{ij} < 0 \end{cases} \tag{14}$$

Then the following is an auxiliary function that upper bounds Eq. 13 [9]:

$$G(\boldsymbol{\alpha}, \tilde{\boldsymbol{\alpha}}) = \mathbf{b}^T\boldsymbol{\alpha} + \frac{1}{2}\sum_i \frac{(\mathbf{A}^+\tilde{\boldsymbol{\alpha}})_i}{\tilde{\alpha}_i}\alpha_i^2 - \frac{1}{2}\sum_{i,j} A_{ij}^-\tilde{\alpha}_i\tilde{\alpha}_j(1 + \ln\frac{\alpha_i\alpha_j}{\tilde{\alpha}_i\tilde{\alpha}_j}). \tag{15}$$

Minimizing Eq. 15 yields the following iterative multiplicative rule with guaranteed convergence to $\boldsymbol{\alpha}^{ML}$:

$$\alpha_i \longleftarrow \alpha_i \frac{-b_i + \sqrt{b_i^2 + 4(\mathbf{A}^+\boldsymbol{\alpha})_i(\mathbf{A}^-\boldsymbol{\alpha})_i}}{2(\mathbf{A}^+\boldsymbol{\alpha})_i}. \tag{16}$$

The iterative formula in Eq. 16 is used to efficiently compute a reasonable estimate for $\boldsymbol{\alpha}^{ML}$ from an arbitrary initialization. However, its convergence is similar to other interior point methods in that small components of $\boldsymbol{\alpha}^{ML}$ will continually decrease but never equal zero. In order to truly sparsify the solution, we employ an alternative method based upon the simplex algorithm for linear programming.

Our other optimization method is based upon finding a solution $\boldsymbol{\alpha}^{ML}$ that satistifies the Karush-Kuhn-Tucker (KKT) conditions for Eq. 13:

$$\mathbf{A}\boldsymbol{\alpha} + \mathbf{b} = \boldsymbol{\beta}, \quad \boldsymbol{\alpha} \geq 0, \boldsymbol{\beta} \geq 0, \alpha_i\beta_i = 0, \quad i = 1, 2, \ldots, M. \tag{17}$$

By introducing additional artificial variables $\mathbf{a}$, the KKT conditions can be transformed into the linear optimization $\min \sum_i a_i$ subject to the constraints:

$$
\begin{align}
\mathbf{a} &\geq 0 \tag{18}\\
\boldsymbol{\alpha} &\geq 0 \tag{19}\\
\boldsymbol{\beta} &\geq 0 \tag{20}\\
\mathbf{A}\boldsymbol{\alpha} - \boldsymbol{\beta} + \operatorname{sign}(-\mathbf{b})\mathbf{a} &= -\mathbf{b} \tag{21}\\
\alpha_i\beta_i &= 0, \quad i = 1, 2, \ldots, M. \tag{22}
\end{align}
$$

The only nonlinear constraint is the product $\alpha_i\beta_i = 0$. However, this can be effectively implemented in the simplex procedure by modifying the selection of the pivot element to ensure that $\alpha_i$ and $\beta_i$ are never both in the set of basic variables. With this simple modification of the simplex algorithm, the optimal $\boldsymbol{\alpha}^{ML}$ can be efficiently computed.

## 2.2 Approximation of $Q(\boldsymbol{\alpha})$

Once the most likely $\boldsymbol{\alpha}^{ML}$ has been determined, the simplest approach for estimating the new $\lambda$ and $\sigma^2$ in Eqs. 9–10 is to replace $Q(\boldsymbol{\alpha}) \approx \delta(\boldsymbol{\alpha} - \boldsymbol{\alpha}^{ML})$ in the integrals. Unfortunately, this simple approximation will cause $\lambda$ and $\sigma$ to diverge from bad initial estimates. To overcome these difficulties, we use a slightly more sophisticated method of estimating the expectations to properly consider variability in the distribution $Q(\boldsymbol{\alpha})$.

We first note that the solution $\boldsymbol{\alpha}^{ML}$ of the nonnegative quadratic optimization in Eq. 12 naturally partitions the elements of the vector $\boldsymbol{\alpha}$ into two distinct subsets $\boldsymbol{\alpha}_I$ and $\boldsymbol{\alpha}_J$, consisting of components $i \in I$ such that $(\boldsymbol{\alpha}^{ML})_i = 0$, and components $j \in J$ such that $(\boldsymbol{\alpha}^{ML})_j > 0$, respectively. It will then be useful to approximate the distribution $Q(\boldsymbol{\alpha})$ as the factored form:

$$
Q(\boldsymbol{\alpha}) \approx Q_I(\boldsymbol{\alpha}_I)Q_J(\boldsymbol{\alpha}_J) \tag{23}
$$

Consider the components $\boldsymbol{\alpha}_J$ near the maximum likelihood solution $\boldsymbol{\alpha}^{ML}$. Among these components, none of nonnegativity constraints are active, so it is reasonable to approximate the distribution $Q_J(\boldsymbol{\alpha}_J)$ by the unconstrained Gaussian:

$$
Q_J(\boldsymbol{\alpha}_J) \propto \exp[-F(\boldsymbol{\alpha}_J|\boldsymbol{\alpha}_I = 0)] \tag{24}
$$

This Gaussian distribution has mean $\boldsymbol{\alpha}_J^{ML}$ and inverse covariance given by the submatrix $\mathbf{A_{JJ}}$ of $\mathbf{A} = \frac{1}{\sigma^2}\mathbf{S}^T\mathbf{S}$.

For the other components $\boldsymbol{\alpha}_I$, it is important to consider the nonnegativity constraints, since $\boldsymbol{\alpha}_I^{ML} = 0$ is on the boundary of the distribution. We can represent $Q_I(\boldsymbol{\alpha}_I)$ with the first two order Tyler expansion:

$$
\begin{align}
Q_I(\boldsymbol{\alpha}_I) &\propto \exp\{-[(\frac{\partial F}{\partial \boldsymbol{\alpha}})|_{\boldsymbol{\alpha}^{ML}}]_I^T \boldsymbol{\alpha}_I - \frac{1}{2}\boldsymbol{\alpha}_I^T \mathbf{A}_{II}\boldsymbol{\alpha}_I)\}, \notag\\
&\propto \exp[-(\mathbf{A}\boldsymbol{\alpha}^{ML} + \mathbf{b})_I^T \boldsymbol{\alpha}_I - \frac{1}{2}\boldsymbol{\alpha}_I^T \mathbf{A}_{II}\boldsymbol{\alpha}_I] \notag\\
&\boldsymbol{\alpha}_I \geq 0. \tag{25}
\end{align}
$$

$Q_I(\boldsymbol{\alpha}_I)$ is then approximated with factorial exponential distribution $\hat{Q}_I(\boldsymbol{\alpha}_I)$ so that the integrals in Eqs. 9–10 can be easily evaluated.

$$
\hat{Q}_I(\boldsymbol{\alpha}_I) = \prod_{i\in I} \frac{1}{\mu_i} e^{-\alpha_i/\mu_i}, \quad \boldsymbol{\alpha}_I \geq 0 \tag{26}
$$

which has support only for nonnegative $\boldsymbol{\alpha}_I \geq 0$. The mean-field parameters $\boldsymbol{\mu}$ are optimally obtained by minimizing the KL-divergence:

$$
\min_{\boldsymbol{\mu}\geq 0} \int_{\boldsymbol{\alpha}_I\geq 0} d\boldsymbol{\alpha}_I\, \hat{Q}_I(\boldsymbol{\alpha}_I) \ln \frac{\hat{Q}_I(\boldsymbol{\alpha}_I)}{Q_I(\boldsymbol{\alpha}_I)}. \tag{27}
$$

This integral can easily be computed in terms of the parameters $\boldsymbol{\mu}$ and yields the minimization:

$$\min_{\boldsymbol{\mu} \geq 0} - \sum_{i \in I} \ln \mu_i + \hat{\mathbf{b}}_I^T \boldsymbol{\mu} + \frac{1}{2} \boldsymbol{\mu}^T \hat{\mathbf{A}} \boldsymbol{\mu}, \tag{28}$$

where $\hat{\mathbf{b}}_I = (\mathbf{A} \boldsymbol{\alpha}^{ML} + \mathbf{b})_I$, $\hat{\mathbf{A}} = \mathbf{A}_{II} + \text{diag}(\mathbf{A}_{II})$. To solve this minimization problem, we use an auxiliary function for Eq. 28 similar to the auxiliary function for nonnegative quadratic programming:

$$G(\boldsymbol{\mu}, \tilde{\boldsymbol{\mu}}) = - \sum_{i \in I} \ln \mu_i + \hat{\mathbf{b}}_I^T \boldsymbol{\mu} + \frac{1}{2} \sum_{i \in I} \frac{(\hat{\mathbf{A}}^+ \tilde{\boldsymbol{\mu}})_i}{\tilde{\mu}_i} \mu_i^2 - \frac{1}{2} \sum_{i,j \in I} \hat{A}_{ij}^- \tilde{\mu}_i \tilde{\mu}_j (1 + \ln \frac{\mu_i \mu_j}{\tilde{\mu}_i \tilde{\mu}_j}), \tag{29}$$

where $\hat{\mathbf{A}} = \hat{\mathbf{A}}^+ - \hat{\mathbf{A}}^-$ is the decomposition of $\hat{\mathbf{A}}$ into its positive and negative components. Minimization of this auxiliary function yields the following multiplicative update rules for $\mu_i$:

$$\mu_i \longleftarrow \mu_i \frac{-\hat{b}_i + \sqrt{\hat{b}_i^2 + 4(\hat{\mathbf{A}}^+ \boldsymbol{\mu})_i [(\hat{\mathbf{A}}^- \boldsymbol{\mu})_i + \frac{1}{\mu_i}]}}{2(\hat{\mathbf{A}}^+ \boldsymbol{\mu})_i}. \tag{30}$$

These iterations are then guaranteed to converge to the optimal mean-field parameters for the distribution $Q_I(\boldsymbol{\alpha}_I)$.

Given the factorized approximation $\hat{Q}_I(\boldsymbol{\alpha}_I) Q_J(\boldsymbol{\alpha}_J)$, the expectations in Eqs. 9–10 can be analytically calculated. The mean value of $\boldsymbol{\alpha}$ under this distribution is given by:

$$\bar{\alpha}_i = \begin{cases} \alpha_i^{ML} & \text{if} \quad i \in J \\ \mu_i & \text{if} \quad i \in I \end{cases} \tag{31}$$

and its covariance $\mathbf{C}$ is:

$$C_{ij} = \begin{cases} (\mathbf{A_{JJ}}^{-1})_{ij} & \text{if} \quad i,j \in J \\ \mu_i^2 \delta_{ij} & \text{otherwise} \end{cases} \tag{32}$$

The update rules for $\lambda$ and $\sigma^2$ are then given by:

$$\lambda \quad \longleftarrow \quad \frac{M}{\sum_i \bar{\alpha}_i} \tag{33}$$

$$\sigma^2 \quad \longleftarrow \quad \frac{1}{N} [(\mathbf{x} - \mathbf{S}\bar{\boldsymbol{\alpha}})^T (\mathbf{x} - \mathbf{S}\bar{\boldsymbol{\alpha}}) + \text{Tr}(\mathbf{S}^T \mathbf{S} \mathbf{C})] \tag{34}$$

To summarize, the complete algorithm consists of the following steps:

1. Initialize $\lambda$ and $\sigma^2$.

2. Determine $\boldsymbol{\alpha}^{ML}$ by solving the nonnegative quadratic programming in Eq. 12.

3. Approximate the distribution $Q(\boldsymbol{\alpha}) \approx \hat{Q}_I(\boldsymbol{\alpha}_I) Q_J(\boldsymbol{\alpha}_J)$ by solving the mean field equations for $\boldsymbol{\mu}$ in $\hat{Q}_I$.

4. Calculate the mean $\bar{\boldsymbol{\alpha}}$ and covariance $\mathbf{C}$ for this distribution.

5. Reestimate regularization parameters $\lambda$ and $\sigma^2$ using Eqs. 33–34.

6. Go back to Step 2 until convergence.

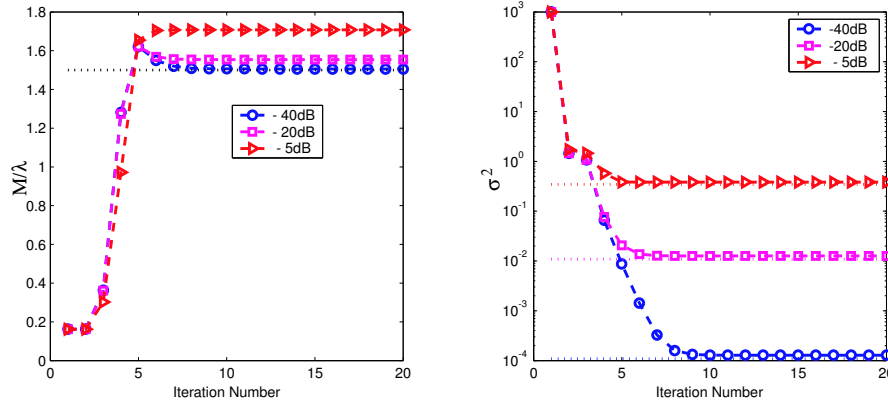

Figure 3: Iterative estimation of $\lambda$ ($M/\lambda$ in the figure, indicating the reverberation level) and $\sigma^2$ when $x(t)$ is contaminated by background white noise at -5 dB, -20 dB, and -40 dB levels. The horizontal dotted lines indicate the true levels.

## 3   Results

We illustrate the performance of our algorithm in estimating the regularization parameters as well as the nonnegative filter coefficients of a speech source signal $s(t)$. The observed signal $x(t)$ is simulated by a time-delayed version of the source signal mixed with an echo along with additive Gaussian white noise $\eta(t)$:

$$x(t) = s(t - T_s) + 0.5s(t - 16.5T_s) + \eta(t). \qquad (35)$$

We compare the results of the algorithm as the noise level is changed. Fig. 3 shows the convergence of the estimates for $\lambda$ and $\sigma^2$ as the noise level is varied between -5 dB and -40 dB. There is rapid convergence of both parameters even with bad initial estimates. The resulting value of the $\sigma^2$ parameter is very close to the true noise level. Additionally, the estimated $\lambda$ parameter is inversely related to the reverberation level of the environment, given by the sum of the true filter coefficients.

Fig. 4 demonstrates the importance of correctly determining the regularization parameters in estimating the time delay structure in the presence of noise. Using the Bayesian regularization procedure, the resulting estimate for $\alpha^{ML}$ correctly models the direct path time delay as well as the secondary echo. However, if the regularization parameters are manually set incorrectly to over-sparsify the solution, the resulting estimates for the time delays may be quite inaccurate.

## 4   Discussion

In summary, we propose using a Bayesian framework to automatically regularize nonnegative deconvolutions for estimating time delays in acoustic signals. We present two methods for efficiently solving the resulting nonnegative quadratic programming problem. We also derive an iterative algorithm from Expectation-Maximization to estimate the regularization parameters. We show how these iterative updates can simultaneously estimate the time-delay structure in the signal, as well as the background noise level and reverberation level of the room. Our results indicate that the algorithm is able to quickly converge to an optimal solution, even with bad initial estimates. Preliminary tests with an acoustic robotic platform indicate that these algorithms can successfully be implemented on a real-time system.

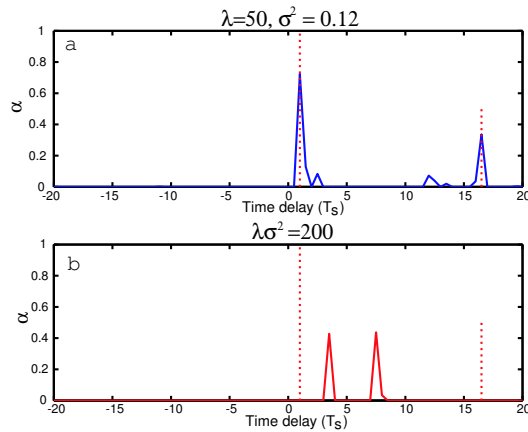

Figure 4: Estimated time delay structure from $\boldsymbol{\alpha}^{ML}$ with different regularizations: a) Bayesian regularization, b) manually set regularization. Dotted lines indicate the true positions of the time delays.

We are currently working to extend the algorithm to the situation where the source signal needs to also be estimated. In this case, priors for the source signal are used to regularize the source estimates. These priors are similar to those used for blind source separation. We are investigating algorithms that can simultaneously estimate the hyperparameters for these priors in addition to the other parameters within a consistent Bayesian framework.

## References

[1] E. Ben-Reuven and Y. Singer, "Discriminative binaural sound localization," in *Advances in Neural Information Processing Systems*, S. T. Suzanna Becker and K. Obermayer, Eds., vol. 15. The MIT Press, 2002.

[2] C. H. Knapp and G. C. Carter., "The generalized correlation method for estimation of time delay," *IEEE Transactions on ASSP*, vol. 24, no. 4, pp. 320–327, 1976.

[3] Y. Lin, D. D. Lee, and L. K. Saul, "Nonnegative deconvolution for time of arrival estimation," in *ICASSP*, 2004.

[4] J. B. Allen and D. A. Berkley, "Image method for efficient simulating small-room acoustics," *J. Acoust. Soc. Am.*, vol. 65, pp. 943–950, 1979.

[5] B. Olshausen and D. Field, "Emergence of simple-cell receptive field properties by learning a sparse code for nature images," *Nature*, vol. 381, pp. 607–609, 1996.

[6] D. Foresee and M. Hagan, "Gauss-Newton approximation to Bayesian regularization," in *Proceedings of the 1997 International Joint Conference on Neural Networks*, 1997, pp. 1930–1935.

[7] M. E. Tipping, "Sparse Bayesian learning and the relevance vector machine," *Journal of Machine Learning Research*, vol. 1, pp. 211–244, 2001.

[8] D. MacKay, "Bayesian interpolation," *Neural Computation*, vol. 4, pp. 415–447, 1992.

[9] F. Sha, L. K. Saul, and D. Lee, "Multiplicative updates for nonnegative quadratic programming in support vector machines," in *Advances in Neural Information Processing Systems*, S. T. Suzanna Becker and K. Obermayer, Eds., vol. 15. The MIT Press, 2002.
